# A Knowledge-Based Model of Geometry Learning

**Geoffrey Towell**
Siemens Corporate Research
755 College Road East
Princeton, NJ 08540

*towell@learning.siemens.com*

**Richard Lehrer**
Educational Psychology
University of Wisconsin
1025 West Johnson St.
Madison, WI 53706
*lehrer@vms.macc.wisc.edu*

## Abstract

We propose a model of the development of geometric reasoning in children that explicitly involves learning. The model uses a neural network that is initialized with an understanding of geometry similar to that of second-grade children. Through the presentation of a series of examples, the model is shown to develop an understanding of geometry similar to that of fifth-grade children who were trained using similar materials.

## 1 Introduction

One of the principal problems in instructing children is to develop sequences of examples that help children acquire useful concepts. In this endeavor it is often useful to have a model of how children learn the material, for a good model can guide an instructor towards particularly effective examples. In short, good models of learning help a teacher maximize the utility of the example presented.

The particular problem with which we are concerned is learning about conventional concepts in geometry, like those involved in identifying, and recognizing similarities and differences among, shapes. This is a difficult subject to teach because children (and adults) have a complex set of informal rules for geometry (that are often at odds with conventional rules). Hence, instruction must supplant this informal geometry with a common formalism. To be efficient in their instruction, teachers need a model of geometric learning which, at the very least:

1. can represent children's understanding of geometry prior to instruction,
2. can describe how understanding changes as a result of instruction,
3. can predict the effect of differing instructional sequences.

In this paper we describe a neural network based model that has these properties.

An extant model of geometry learning, the "van Hiele model" [6] represents children's understanding as purely perceptual -- appearances dominate reasoning. However, our research suggests that children's reasoning is better characterized as a mix of perception and rules. Moreover, unlike the model we propose, the van Hiele model can neither be used to test the effectiveness of instruction prior to trying that instruction on children nor can it be used to describe how understanding changes as a result of a specific type of instruction.

Briefly, our model uses a set of rules derived from interviews with first and second grade children [1, 2], to produce a stereotypical informal conception of geometry. These rules, described in more detail in Section 2.1, give our model an explicit representation of pre-instructional geometry understanding. The rules are then translated into a neural network using the KBANN algorithm [3]. As a neural network, our model can test the effect of differing instructional sequences by simply training two instances with different sets of examples. The experiments in Section 3 take advantage of this ability of our model; they show that it is able to accurately model the effect of two different sets of instruction.

## 2   A New Model

This section describes the initial state of our model and its implementation as a neural network. The initial state of the model is intended to reproduce the decision processes of a typical child prior to instruction. The methodology used to derive this information and a brief description of this information both are in the first subsection. In addition, this subsection contains a small experiment that shows the accuracy of the initial state of the model. In the next subsection, we briefly describe the translation of those rules into a neural network.

### 2.1   The initial state of the model

Our model is based upon interviews with children in first and second grade [1, 2]. In these interviews, children were presented with sets of three figures such as the triad in Figure 1. They were asked which pair of the three figures is the most similar and why they made their decision. These interviews revealed that, prior to instruction, children base judgments of similarity upon the seven attributes in Table 1.

For the triad discrimination task, children find ways in which a pair is similar that is not shared by the other two pairs. For instance, B and C in Figure 1.2 are both *pointy* but A is not. As a result, the modal response of children prior to instruction is that {B C} is the most similar pair. This decision making process is described by the rules in Table 2.

In addition to the rules in Table 2, we include in our initial model a set of rules that describe templates for standard geometric shapes. This addition is based upon interviews with children which suggest that they know the names of shapes such as triangles and squares, and that they associate with each name a small set of templates. Initially, children treat these shape names as having no more importance than any of the attributes in Table 1. So, our model initial treats shape names exactly as one of those attributes. Over time children learn that the names of shapes are very important because they are diagnostic (the name indicates properties). Our hope was that the model would make a similar transition so that the shape names would become sufficient for similarity determination.

Note that the rules in Table 2 do not always yield a unique decision. Rather, there are

Table 1: Attributes used by children prior to instruction.

| Attribute name | Possible values | Attribute name | Possible values |
|---|---|---|---|
| Tilt | 0, 10, 20, 30, 40 | Slant | yes, no |
| Area | small, medium, large | Shape | skinny, medium, fat |
| Pointy | yes, no | Direction | ←, →, ↑, ↓ |
| 2 long & short | yes, no | | |

Table 2: Rules for similarity judgment in the triad discrimination task.

```
1. IF fig-val(fig1?, att?) = fig-val(fig2?, att?) THEN
   same-att-value(fig1?, fig2?, att?).
2. IF not(same-att-value(fig1?, fig3?, att?)) AND fig1? ≠ fig3?
   AND fig2? ≠ fig3? THEN unq-sim(fig1?, fig2?, att?).
3. IF c(unq-sim(fig1?, fig2?, att?)) >
   c(unq-sim(fig1?, fig2?, att?)) AND
   c(unq-sim(fig1?, fig3?, att?)) > c(unq-sim(fig2?, fig3?, att?))
   AND fig1? ≠ fig3? AND fig2?≠ fig3? THEN
   most-similar(fig1?, fig2?).
```

Labels followed by a '?' indicate variables.
fig-val(fig?, att?) returns the value of att? in fig?
c() counts the number of instances.

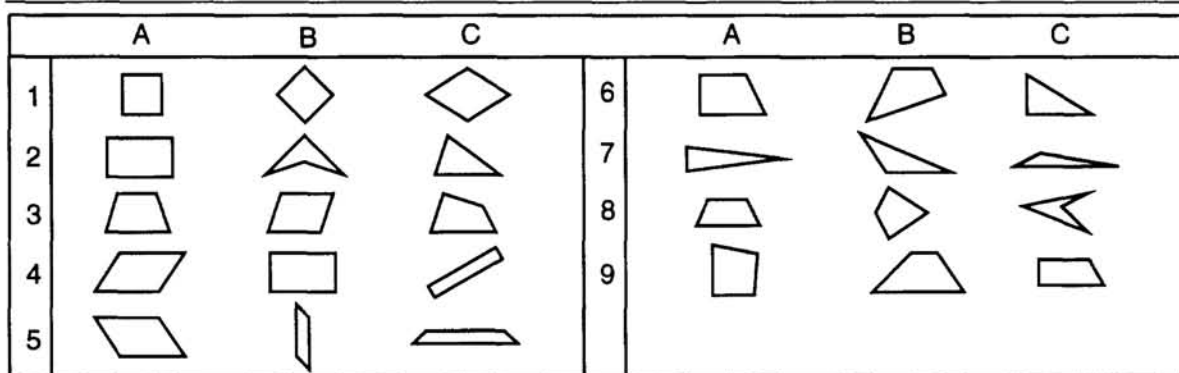

Figure 1: Triads used to test learning.

triads for which these rules cannot decide which pair is most similar. This is not often the case for a particular child, who usually finds one attribute more salient than another. Yet, frequently when the rules cannot uniquely identify the most similar pair, a classroom of children is equally divided. Hence, the model may not accurately predict an individual response, but is it usually correct at identifying the modal responses.

To verify the accuracy of the initial state of our model, we used the set of nine testing triads shown in Figure 1 which were developed for the interviews with children. As shown in Table 3, the model matches very nicely responses obtained from a separate sample of 48 second grade children. Thus, we believe that we have a valid point from which to start.

## 2.2   The translation of rule sets into a neural network

We translate rules sets into neural networks using the KBANN algorithm [3] which uses a set of hierarchically-structured rules in the form of propositional Horn clauses to set the topology and initial weights of an artificial neural network. Because the rules in Table 2 are

Table 3: Initial responses by the model.

| Triad Number | 1 | 2 | 3 | 4 | 5 | 6 | 7 | 8 | 9 |
|---|---|---|---|---|---|---|---|---|---|
| Initial Model | BC | BC | AC | AC | BC | AB/BC | AC | **AB/BC** | AC/BC |
| Second Grade Children | BC | BC | AC | AC | BC | AB/BC | AC | **AB** | AC/BC |

Answers in the "initial model" row indicate the responses generated by the initial rules. More than response in a column indicates that the rules could not differentiate among two pairs.

Answers in the "second grade" row are the modal responses of second grade children. More than one answer in a column indicates that equal numbers of children judged the pairs most similar.

Table 4: Properties used to describe figures.

| Property name | values | Property name | values |
|---|---|---|---|
| Convex | Yes No | # Pairs Equal Opposite Angles | 0 1 2 3 4 |
| # Sides | 3 4 5 6 8 | # Pairs Opposite Sides Equal | 0 1 2 3 4 |
| # Angles | 3 4 5 6 8 | # Pairs Parallel Sides | 0 1 2 3 4 |
| All Sides Equal | Yes No | Adjacent Angles = 180 | Yes No |
| # Right Angles | 0 1 2 3 4 | # Lines of Symmetry | 0 1 2 3 4 5 6 8 |
| All Angles Equal | Yes No | # Equal Sides | 0 2 3 4 5 6 8 |
| # Equal Angles | 0 2 3 4 5 6 8 | | |

not in propositional form, they must be expanded before they can be accepted by KBANN. The expansion turns a simple set of three rules into an ugly set of approximately 100 rules.

Figure 2 is a high-level view of the structure of the neural network that results from the rules. In this implementation we present all three figures at the same time and all decisions are made in parallel. Hence, the rules described above must be repeated at least three times. In the neural network that results from the rule translation, these repeated rules are not independent. Rather they are linked so that modifications of the rules are shared across every pairing. Thus, the network cannot learn a rule which applies only to one pair.

Finally, the model begins with the set of 13 properties listed in Table 4 in addition to the attributes of Table 1. (Note that we use "attribute" to refer to the informal, visual features in Table 1 and "property" to refer to the symbolic features in Table 4.) As a result, each figure is described to the model as a 74 position vector (18 positions encode the attributes; the remaining 56 positions encode the properties).

## 3    An Experiment Using the Model

One of the points we made in the introduction is that a useful model of geometry learning should be able to predict the effect of instruction. The experiment reported in this section tests this facet of our model. Briefly, this experiment trains two instances of our model using different sets of data. We then compare the instances to children who have been trained using a set of problems similar to one of those used to train the model. Our results show that the two instances learn quite different things. Moreover, the instance trained witn material similar to the children predicts the children's responses on test problems with a high level of accuracy.

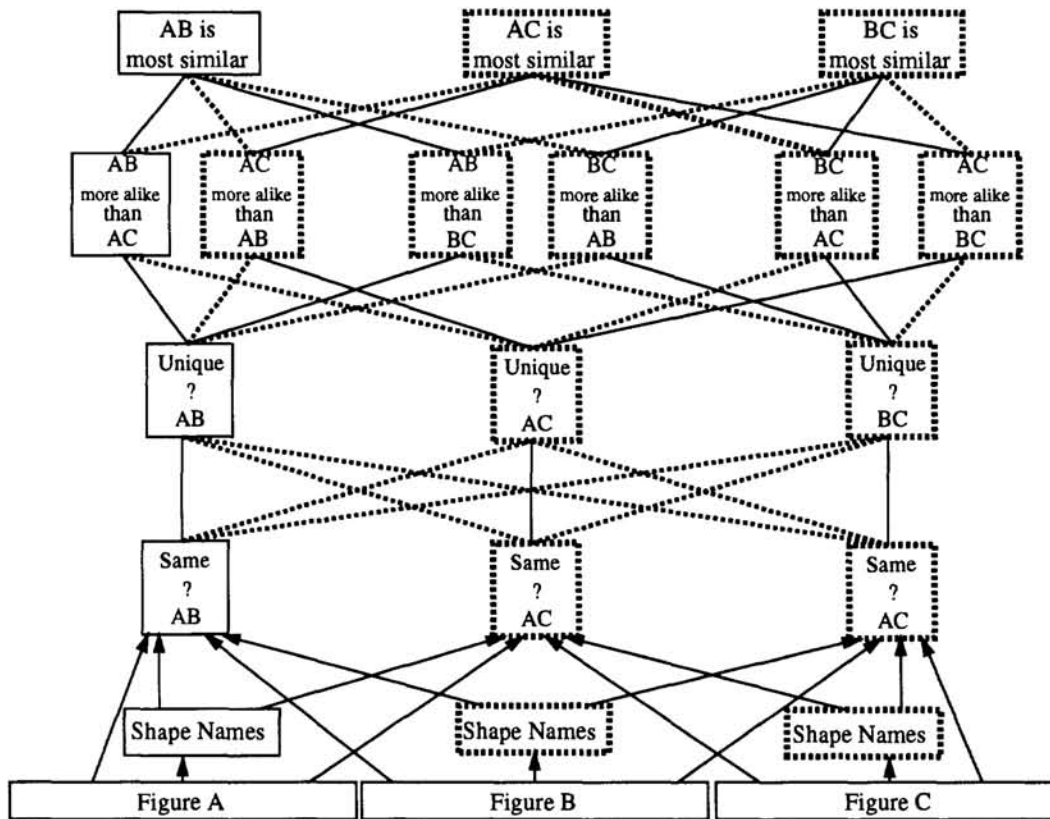

Boxes indicate one or more units.
Dashed boxes indicate units associated with duplicated rules
Dashed lines indicate one or more negatively weighted links.
Solid lines indicate one or more positively weighted links.

Figure 2: The structure of the neural network for our model.

## 3.1  Training the model

For this experiment, we developed two sets of training shapes. One set contains every polygon in a fifth-grade math textbook [4] (Figure 3). The other set consists of 81 items which might be produced by a child using a modified version of LOGO (Figure 4). Here we assume that one of the effects of learning geometry with a tool like LOGO is simply to increase the extent and range of possible examples. A collection of 33 triads were selected from each set to train the model.[1] Training consisted of repeated presentations of each of the 33 triads until the network correctly identified the most similar pair for each triad.

## 3.2  Tests of the model

In this section, we test the ability of the model to accurately predict the effects of instruction. We do this by comparing the two trained instances of the model to the modal responses of fifth graders who had used LOGO for two weeks. In those two weeks, the children had generates many (but not all) of the figures in Figure 4. Hence, we expected that the instance

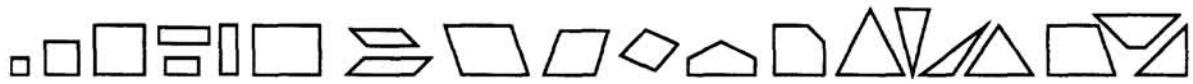

Figure 3: Representative textbook shapes.

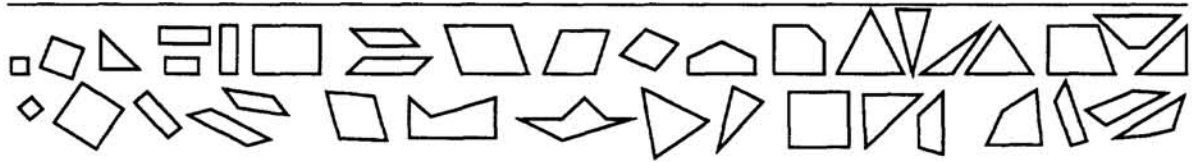

Figure 4: Representative shapes encountered using a modified version of LOGO.

of the model trained using triads drawn from Figure 4 would better predict the responses of these children than the other instance of the model.

Clearly, the results in Table 5 verify our expectations. The LOGO-trained model agrees with the modal responses of children on an average of six examples while the textbook-trained model agrees on an average of three examples. The respective binomial probabilities of six and three matches is 0.024 and 0.249. These probabilities suggest that the match between the LOGO-trained model and the children is unlikely to have occurred by chance. On the other hand, the instance of the model trained by the textbook examples has the most probable outcome from simply random guessing. Thus, we conclude that the LOGO-trained model is a good predictor of children's learning when using LOGO.

In addition, whereas the textbook-trained model was no better than chance at estimating the conventional response, the LOGO-trained model matched convention on an average of seven triads. Interestingly, on both triads where the LOGO-trained model did not match convention, it could not due to lack of appropriate information. For triad 3, convention matches the trapezoid with the parallelogram rather than either of these with the quadrilateral because the trapezoid and the parallelogram both have some pairs of parallel lines. The model, however, has only information about the number of pairs of parallel lines. On the basis of this feature, the three figures are equally dissimilar. For triad 7, the other triad for which the LOGO-trained model did not match convention, the conventional paring matches two obtuse triangles. However, the model has no information about angles other than number and number of right angles. Hence, it could not possibly get this triad correct (at least not for the right reason). We expect that correcting these minor weaknesses will improve the model's ability to make the conventional response.

Table 5: Responses after learning by trained instances of the model and children.

| Triad Number | 1 | 2 | 3 | 4 | 5 | 6 | 7 | 8 | 9 |
|---|---|---|---|---|---|---|---|---|---|
| Textbook Trained | AB/BC | BC | AC | AC | BC | AB | AC | AB | AC |
| LOGO Trained | AB/BC | AB | ?? | BC | AB | AB | AB | AB | BC |
| Fifth Grade Children | AB/BC | AB | AC/AB | BC | AB | AB/BC | AC | AB/BC | BC |
| Convention | AB | AB | AB | BC | AB | AB | BC | AB | BC |

Responses by the model are the modal responses over 500 trials.
?? indicates that the model was unable to select among the pairings.

The success of our model in the prediction experiment lead us to investigate the reasons underlying the answers generated by its two instances. In so doing we hoped to gain an understanding of the networks' reasoning processes. Such an understanding would

be invaluable in the design of instruction for it would allow the selection of examples that fill specific learning deficits. Unfortunately, trained neural networks are often nearly impossible to comprehend. However, using tools such as those described by Towell and Shavlik [5], we believe that we developed a reasonably clear understanding of the effects of each set of training examples.

The LOGO-trained model made comprehensive adjustments of its initial conditions. Of the eight attributes, it attends to only size and 2 long & short after training. While learning to ignore most of the attributes, the model also learned to pay attention to several of the properties. In particular, number of angles, number of sides, all angles, equal, all sides equal, and number of pairs of opposite sides parallel, all were important to the network after training. Thus, the LOGO-trained instance of the model made a significant transition in its basis for geometric reasoning. Sadly, in making this transition, the declarative clarity of the initial rules was lost. Hence, it is impossible to precisely state the rules that the trained model used to make its final decisions.

By contrast, the textbook-trained instance of the model failed to learn that most of the attributes were unimportant. Instead, the model simply learned that several of the properties were also important. As a result, reasons for answers on the test set often seemed schizophrenic. For instance, in responding BC on test triad 2, the network attributed the decision to similarities in: area, pointiness, point-direction, number of sides, number of angles, number of right angle and all angles equal. Given this combination, it is not surprising that the example is answered incorrectly. This result suggests that typical textbooks may accentuate the importance of conventional properties, but they provide little grist for abandoning the mill of informal attributes.

### 3.3  Discussion

This experiment demonstrated the utility of our model in several ways. First, it showed that the model is sensitive to differences in training set. Of itself, this is neither a surprising nor interesting conclusion. What is important about the difference in learning is that the model trained in a manner similar to a classroom of fifth grade children made responses to the test set that we quite similar to those of fifth grade children.

In addition to making different responses to the test set, the two trained instances of the model appeared to learn different things. In particular, the LOGO-trained instance essentially replaced its initial knowledge with something much more like the formal geometry. On the other hand, the textbook-trained instance simply added several concepts from formal geometry to the informal concept with which it was initialized. An improved transition from informal to formal geometry is one of the advantages claimed for LOGO based instruction [2]. Hence, the difference between the two instances of the model agrees with observation of children.

This result suggests that our model is able to predict the effect of differing instructional sequences. A further experiment of this hypothesis would be to use our model to design a set of instruction materials. This could be done by starting with an apparently good set of materials, training the model, examining its deficiencies and revising the training materials appropriately. Our hypothesis is that a set of materials so constructed would be superior to the materials normally used in classrooms. Testing of this hypothesis is one of our major directions for future research.

## 4  Conclusions

In this paper we have described a model of the initial stages of geometry learning by elementary school children. This model is initialized using a set of rules based upon interviews with first and second grade children. This set of rules is shown to accurately predict the responses of second grade children on a hard set of similarity determination problems.

Given that we have a valid starting point for our model, we test it by training those rules, after re-representing them in a neural network, with two different sets of training materials. Each instance of the model is analyzed in two ways. First, they are compared, on an independent set of testing examples, to fifth grade children who had been trained using materials similar to one of the model's training sets. This comparison showed that the model trained with materials similar to the children accurately reproduced the responses of the children. The second analysis involved examining the model after training to determine what it had learned. Both instances of the model learned to attend to the properties that were not mentioned in the initial rules. The model trained with the richer (LOGO-based) training set also learned that the informal attributes were relatively unimportant. Conversely, the model trained with the textbook-based training examples merely added information about properties to the pre-existing information. Therefore, we believe that the model we have described is has the potential to become a valuable tool for teachers.

## Footnotes

[1] In choosing the same number of triads for each training set, we are being very generous to the textbook. In reality, not only do children see more figures when using LOGO, they are also able to make many more contrasts between figures. Hence, it might be more accurate to make the LOGO training set much larger than the textbook training set.

## References

[1] R. Lehrer, W. Knight, M. Love, and L. Sancilio. Software to link action and description in pre-proof geometry. Presented at the Annual Meeting of the American Educational Research Association, 1989.

[2] R. Lehrer, L. Randle, and L. Sancilio. Learning preproof geometry with LOGO. *Cognition and Instruction*, 6:159--184, 1989.

[3] M. O. Noordewier, G. G. Towell, and J. W. Shavlik. Training knowledge-based neural networks to recognize genes in DNA sequences. In *Advances in Neural Information Processing Systems*, volume 3, pages 530--536, Denver, CO, 1991. Morgan Kaufmann.

[4] M. A. Sobel, editor. *Mathematics*. McGraw-Hill, New York, 1987.

[5] G. G. Towell and J. W. Shavlik. Interpretation of artificial neural networks: Mapping knowledge-based neural networks into rules. In *Advances in Neural Information Processing Systems*, volume 4, pages 977--984, Denver, CO, 1991. Morgan Kaufmann.

[6] P. M. van Hiele. *Structure and Insight*. Academic Press, New York, 1986.
